# Robot Learning: Exploration and Continuous Domains

**David A. Cohn**
MIT Dept. of Brain and Cognitive Sciences
Cambridge, MA 02139

The goal of this workshop was to discuss two major issues: efficient exploration of a learner's state space, and learning in continuous domains. The common themes that emerged in presentations and in discussion were the importance of choosing one's domain assumptions carefully, mixing controllers/strategies, avoidance of catastrophic failure, new approaches with difficulties with reinforcement learning, and the importance of task transfer.

## 1  Domain assumptions

Andrew Moore (CMU) discussed the problem of standardizing and making explicit the set of assumptions that researcher makes about his/her domain. He suggested that neither "fewer assumptions are better" nor "more assumptions are better" is a tenable position, and that we should strive to find and use standard sets of assumptions. With no such commonality, comparison of techniques and results is meaningless. Under Moore's guidance, the group discussed the possibility of designing an algorithm which used a number of well-chosen assumption sets and switched between them according to their empirical validity. Suggestions were made to draw on the AI approach of truth maintenance systems.

This theme of detecting failure of an assumption set/strategy was echoed in the discussion on mixing controllers and avoiding failure (described below).

## 2  Mixing controllers and strategies

Consensus appeared to be against using single monolithic approaches, and in favor of mixing controllers. Spatial mixing resulted in local models, as advocated by Stefan Schaal (MIT) using locally weighted regression.

Controllers could also be mixed over the entire domain. Jeff Schneider (Rochester) discussed mixing a naïve feedback controller with a "coaching signal." Combining the coached controller with an uncoached one further improved performance. During the main conference, Satinder Singh (MIT) described a controller that learned by reinforcement to mix the strategies of two "safe" but suboptimal controllers, thus avoiding unpleasant surprises and catastrophic failure.

## 3 Avoiding failure

The issue of detecting impending failure and avoiding catastrophic failure clarified differences in several approaches. A learning strategy that learns in few trials is useless on a real robot if the initial trials break the robot by crashing it into walls. Singh's approach has implicit failure avoidance, but may be hampered by an unnecessarily large margin of safety. Terry Sanger (JPL) discussed a trajectory extension algorithm by which a controller could smoothly "push the limits" of its performance, and detect impending failure of the control strategy.

## 4 Reinforcement learning

Reinforcement learning seems to have come into its own, with people realizing the diverse ways in which it may be applied to problems. Long-Ji Lin (Siemens) described his group's unusual but successful application of reinforcement learning in landmark-based navigation. The Siemens RatBot uses reinforcement to select landmarks on the basis of their recognizability and their value to the eventual precision of position estimation. The reinforcement signal is simply the cost and the robot's final position error after it has used a set of landmarks.

José del R. Millán (JRC) presented an approach similar to Schneider's, but training a neural controller with reinforcement learning. As the controller's performance improves, it supplants the mobile robot's reactive "instincts," which are designed to prevent catastrophic failure.

With new applications, however, come new pitfalls. Leemon Baird (WPAFB) showed how standard reinforcement learning approaches can fail when adapted to exploration in continuous time. He then described the "advantage updating" algorithm which was designed to work in noisy domains with continuous or small time steps. The issue of exploration in continuous space, especially with noise, has not been as easily addressed. Jürgen Schmidhuber (TUM) described the approach one should take if interested solely in exploration: use prior information gain as a reinforcement signal to decide on an "optimal" action. The ensuing discussion centered on the age-old and intractable tradeoff between exploration and exploitation. Final consensus was that we, as a group, should become more familiar with the literature on dual control, which addresses exactly this issue.

## 5 Task transfer

An unorchestrated theme that emerged from the discussion was the need to address, or even define, task transfer. As with last year's workshop on Robot Learning, it was generally agreed that "one-task learning" is not a suitable goal when designing a learning robot. During the discussion, Long-Ji Lin (Siemens) and Lori Pratt (CSM) described several types of task transfer that are considered in the literature. These included model learning for multiple tasks, hierarchical control and learning, and concept (or bias) sharing across tasks.